# Bayesian Surprise Attracts Human Attention

**Laurent Itti**
Department of Computer Science
University of Southern California
Los Angeles, California 90089-2520, USA
itti@usc.edu

**Pierre Baldi**
Department of Computer Science
University of California, Irvine
Irvine, California 92697-3425, USA
pfbaldi@ics.uci.edu

## Abstract

The concept of surprise is central to sensory processing, adaptation, learning, and attention. Yet, no widely-accepted mathematical theory currently exists to quantitatively characterize surprise elicited by a stimulus or event, for observers that range from single neurons to complex natural or engineered systems. We describe a formal Bayesian definition of surprise that is the only consistent formulation under minimal axiomatic assumptions. Surprise quantifies how data affects a natural or artificial observer, by measuring the difference between posterior and prior beliefs of the observer. Using this framework we measure the extent to which humans direct their gaze towards surprising items while watching television and video games. We find that subjects are strongly attracted towards surprising locations, with 72% of all human gaze shifts directed towards locations more surprising than the average, a figure which rises to 84% when considering only gaze targets simultaneously selected by all subjects. The resulting theory of surprise is applicable across different spatio-temporal scales, modalities, and levels of abstraction.

Life is full of surprises, ranging from a great christmas gift or a new magic trick, to wardrobe malfunctions, reckless drivers, terrorist attacks, and tsunami waves. Key to survival is our ability to rapidly attend to, identify, and learn from surprising events, to decide on present and future courses of action [1]. Yet, little theoretical and computational understanding exists of the very essence of surprise, as evidenced by the absence from our everyday vocabulary of a quantitative unit of surprise: Qualities such as the "wow factor" have remained vague and elusive to mathematical analysis.

Informal correlates of surprise exist at nearly all stages of neural processing. In sensory neuroscience, it has been suggested that only the unexpected at one stage is transmitted to the next stage [2]. Hence, sensory cortex may have evolved to adapt to, to predict, and to quiet down the expected statistical regularities of the world [3, 4, 5, 6], focusing instead on events that are unpredictable or surprising. Electrophysiological evidence for this early sensory emphasis onto surprising stimuli exists from studies of adaptation in visual [7, 8, 4, 9], olfactory [10, 11], and auditory cortices [12], subcortical structures like the LGN [13], and even retinal ganglion cells [14, 15] and cochlear hair cells [16]: neural response greatly attenuates with repeated or prolonged exposure to an initially novel stimulus. Surprise and novelty are also central to learning and memory formation [1], to the point that surprise is believed to be a necessary trigger for associative learning [17, 18],

as supported by mounting evidence for a role of the hippocampus as a novelty detector [19, 20, 21]. Finally, seeking novelty is a well-identified human character trait, with possible association with the dopamine D4 receptor gene [22, 23, 24].

In the Bayesian framework, we develop the only consistent theory of surprise, in terms of the difference between the posterior and prior distributions of beliefs of an observer over the available class of models or hypotheses about the world. We show that this definition derived from first principles presents key advantages over more *ad-hoc* formulations, typically relying on detecting outlier stimuli. Armed with this new framework, we provide direct experimental evidence that surprise best characterizes what attracts human gaze in large amounts of natural video stimuli. We here extend a recent pilot study [25], adding more comprehensive theory, large-scale human data collection, and additional analysis.

## 1 Theory

**Bayesian Definition of Surprise.** We propose that surprise is a general concept, which can be derived from first principles and formalized across spatio-temporal scales, sensory modalities, and, more generally, data types and data sources. Two elements are essential for a principled definition of surprise. First, surprise can exist only in the presence of uncertainty, which can arise from intrinsic stochasticity, missing information, or limited computing resources. A world that is purely deterministic and predictable in real-time for a given observer contains no surprises. Second, surprise can only be defined in a relative, subjective, manner and is related to the expectations of the observer, be it a single synapse, neuronal circuit, organism, or computer device. The same data may carry different amount of surprise for different observers, or even for the same observer taken at different times.

In probability and decision theory it can be shown that the only consistent and optimal way for modeling and reasoning about uncertainty is provided by the Bayesian theory of probability [26, 27, 28]. Furthermore, in the Bayesian framework, probabilities correspond to subjective degrees of beliefs in hypotheses or models which are updated, as data is acquired, using Bayes' theorem as the fundamental tool for transforming prior belief distributions into posterior belief distributions. Therefore, within the same optimal framework, the only consistent definition of surprise must involve: (1) probabilistic concepts to cope with uncertainty; and (2) prior and posterior distributions to capture subjective expectations.

Consistently with this Bayesian approach, the background information of an observer is captured by his/her/its prior probability distribution $\{P(M)\}_{M \in \mathcal{M}}$ over the hypotheses or models $M$ in a model space $\mathcal{M}$. Given this prior distribution of beliefs, the fundamental effect of a new data observation $D$ on the observer is to change the prior distribution $\{P(M)\}_{M \in \mathcal{M}}$ into the posterior distribution $\{P(M|D)\}_{M \in \mathcal{M}}$ via Bayes theorem, whereby

$$\forall M \in \mathcal{M}, \qquad P(M|D) = \frac{P(D|M)}{P(D)} P(M). \qquad (1)$$

In this framework, the new data observation $D$ carries no surprise if it leaves the observer beliefs unaffected, that is, if the posterior is identical to the prior; conversely, $D$ is surprising if the posterior distribution resulting from observing $D$ significantly differs from the prior distribution. Therefore we formally measure surprise elicited by data as some distance measure between the posterior and prior distributions. This is best done using the relative entropy or Kullback-Leibler $(KL)$ divergence [29]. Thus, surprise is defined by the average of the log-odd ratio:

$$S(D, \mathcal{M}) = KL(P(M|D), P(M)) = \int_{\mathcal{M}} P(M|D) \log \frac{P(M|D)}{P(M)} dM \qquad (2)$$

taken with respect to the posterior distribution over the model class $\mathcal{M}$. Note that $KL$ is not symmetric but has well-known theoretical advantages, including invariance with respect to

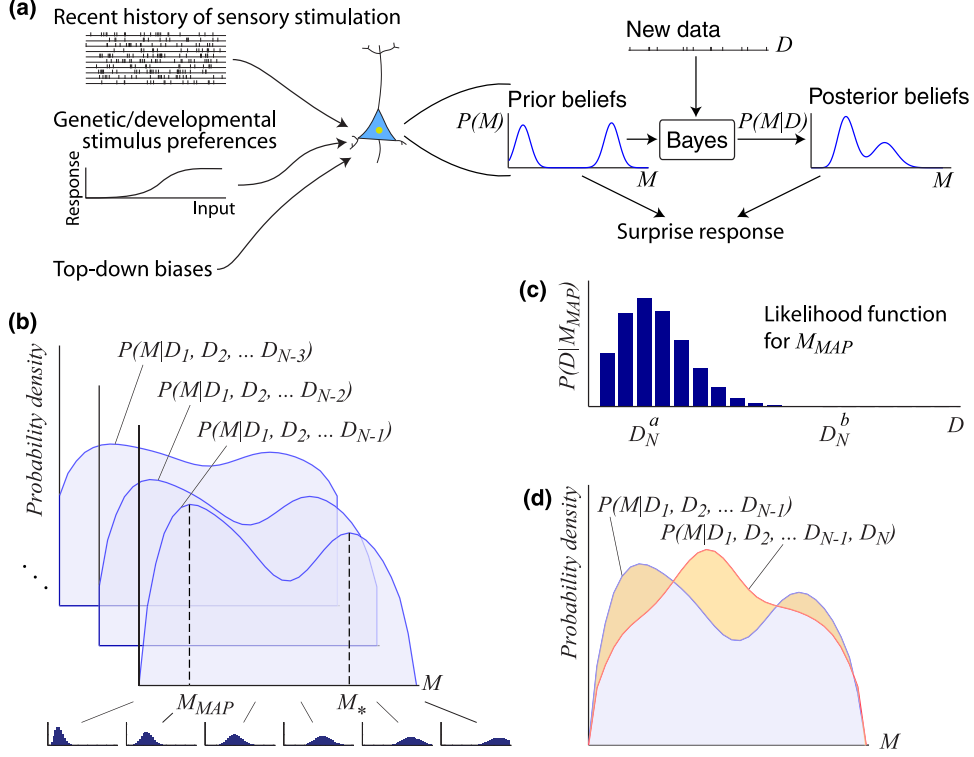

Figure 1: Computing surprise in early sensory neurons. **(a)** Prior data observations, tuning preferences, and top-down influences contribute to shaping a set of "prior beliefs" a neuron may have over a class of internal models or hypotheses about the world. For instance, $\mathcal{M}$ may be a set of Poisson processes parameterized by the rate $\lambda$, with $\{P(M)\}_{M \in \mathcal{M}} = \{P(\lambda)\}_{\lambda \in \mathbb{R}^{+*}}$ the prior distribution of beliefs about which Poisson models well describe the world as sensed by the neuron. New data $D$ updates the prior into the posterior using Bayes' theorem. Surprise quantifies the difference between the posterior and prior distributions over the model class $\mathcal{M}$. The remaining panels detail how surprise differs from conventional model fitting and outlier-based novelty. **(b)** In standard iterative Bayesian model fitting, at every iteration $N$, incoming data $D_N$ is used to update the prior $\{P(M|D_1, D_2, ..., D_{N-1})\}_{M \in \mathcal{M}}$ into the posterior $\{P(M|D_1, D_2, ..., D_N)\}_{M \in \mathcal{M}}$. Freezing this learning at a given iteration, one then picks the currently best model, usually using either a maximum likelihood criterion, or a maximum a posteriori one (yielding $M_{MAP}$ shown). **(c)** This best model is used for a number of tasks at the current iteration, including outlier-based novelty detection. New data is then considered novel at that instant if it has low likelihood for the best model (e.g., $D_N^b$ is more novel than $D_N^a$). This focus onto the single best model presents obvious limitations, especially in situations where other models are nearly as good (e.g., $M_*$ in panel (b) is entirely ignored during standard novelty computation). One palliative solution is to consider mixture models, or simply $P(D)$, but this just amounts to shifting the problem into a different model class. **(d)** Surprise directly addresses this problem by simultaneously considering all models and by measuring how data changes the observer's distribution of beliefs from $\{P(M|D_1, D_2, ..., D_{N-1})\}_{M \in \mathcal{M}}$ to $\{P(M|D_1, D_2, ..., D_N)\}_{M \in \mathcal{M}}$ over the entire model class $\mathcal{M}$ (orange shaded area).

reparameterizations. A unit of surprise — a *"wow"* — may then be defined for a single model $M$ as the amount of surprise corresponding to a two-fold variation between $P(M|D)$ and $P(M)$, i.e., as $\log P(M|D)/P(M)$ (with $\log$ taken in base 2), with the total number of wows experienced for all models obtained through the integration in eq. 2.

**Surprise and outlier detection.** Outlier detection based on the likelihood $P(D|M_{\text{best}})$ of $D$ given a single best model $M_{\text{best}}$ is at best an approximation to surprise and, in some

cases, is misleading. Consider, for instance, a case where $D$ has very small probability both for a model or hypothesis $M$ and for a single alternative hypothesis $\overline{M}$. Although $D$ is a strong outlier, it carries very little information regarding whether $M$ or $\overline{M}$ is the better model, and therefore very little surprise. Thus an outlier detection method would strongly focus attentional resources onto $D$, although $D$ is a false positive, in the sense that it carries no useful information for discriminating between the two alternative hypotheses $M$ and $\overline{M}$. Figure 1 further illustrates this disconnect between outlier detection and surprise.

## 2  Human experiments

To test the surprise hypothesis — that surprise attracts human attention and gaze in natural scenes — we recorded eye movements from eight naïve observers (three females and five males, ages 23-32, normal or corrected-to-normal vision). Each watched a subset from 50 videoclips totaling over 25 minutes of playtime (46,489 video frames, $640 \times 480$, 60.27 Hz, mean screen luminance 30 cd/m$^2$, room 4 cd/m$^2$, viewing distance 80cm, field of view $28° \times 21°$). Clips comprised outdoors daytime and nighttime scenes of crowded environments, video games, and television broadcast including news, sports, and commercials. Right-eye position was tracked with a 240 Hz video-based device (ISCAN RK-464), with methods as previously [30]. Two hundred calibrated eye movement traces (10,192 saccades) were analyzed, corresponding to four distinct observers for each of the 50 clips. Figure 2 shows sample scanpaths for one videoclip.

To characterize image regions selected by participants, we process videoclips through computational metrics that output a topographic dynamic master response map, assigning in real-time a response value to every input location. A good master map would highlight, more than expected by chance, locations gazed to by observers. To score each metric we hence sample, at onset of every human saccade, master map activity around the saccade's future endpoint, and around a uniformly random endpoint (random sampling was repeated 100 times to evaluate variability). We quantify differences between histograms of master

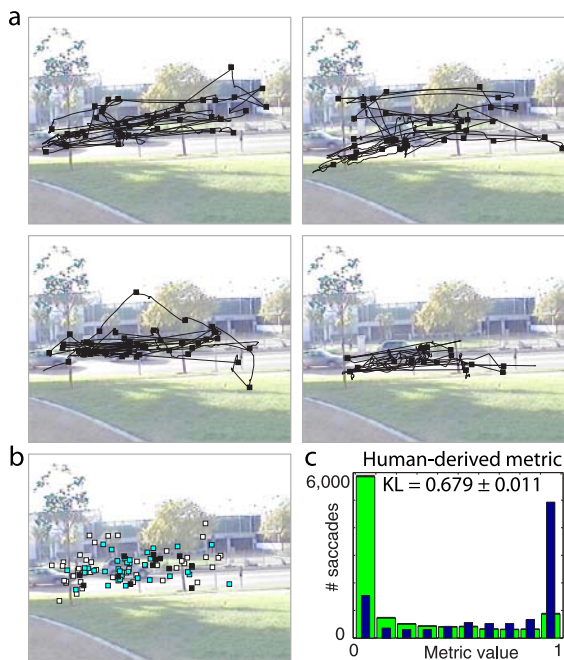

Figure 2: **(a)** Sample eye movement traces from four observers (squares denote saccade endpoints). **(b)** Our data exhibits high inter-individual overlap, shown here with the locations where one human saccade endpoint was nearby ($\approx 5°$) one (white squares), two (cyan squares), or all three (black squares) other humans. **(c)** A metric where the master map was created from the three eye movement traces other than that being tested yields an upper-bound $KL$ score, computed by comparing the histograms of metric values at human (narrow blue bars) and random (wider green bars) saccade targets. Indeed, this metric's map was very sparse (many random saccades landing on locations with near-zero response), yet humans preferentially saccaded towards the three active hotspots corresponding to the eye positions of three other humans (many human saccades landing on locations with near-unity responses).

map samples collected from human and random saccades using again the Kullback-Leibler ($KL$) distance: metrics which better predict human scanpaths exhibit higher distances from random as, typically, observers non-uniformly gaze towards a minority of regions with highest metric responses while avoiding a majority of regions with low metric responses. This approach presents several advantages over simpler scoring schemes [31, 32], including agnosticity to putative mechanisms for generating saccades and the fact that applying any continuous nonlinearity to master map values would not affect scoring.

**Experimental results.** We test six computational metrics, encompassing and extending the state-of-the-art found in previous studies. The first three quantify static image properties (local intensity variance in $16 \times 16$ image patches [31]; local oriented edge density as measured with Gabor filters [33]; and local Shannon entropy in $16 \times 16$ image patches [34]). The remaining three metrics are more sensitive to dynamic events (local motion [33]; outlier-based saliency [33]; and surprise [25]).

For all metrics, we find that humans are significantly attracted by image regions with higher metric responses. However, the static metrics typically respond vigorously at numerous visual locations (Figure 3), hence they are poorly specific and yield relatively low $KL$ scores between humans and random. The metrics sensitive to motion, outliers, and surprising events, in comparison, yield sparser maps and higher $KL$ scores.

The surprise metric of interest here quantifies low-level surprise in image patches over space and time, and at this point does not account for high-level or cognitive beliefs of our human observers. Rather, it assumes a family of simple models for image patches, each processed through 72 early feature detectors sensitive to color, orientation, motion, etc., and computes surprise from shifts in the distribution of beliefs about which models better describe the patches (see [25] and [35] for details). We find that the surprise metric significantly outperforms all other computational metrics ($p < 10^{-100}$ or better on $t$-tests for equality of $KL$ scores), scoring nearly 20% better than the second-best metric (saliency) and 60% better than the best static metric (entropy). Surprising stimuli often substantially differ from simple feature outliers; for example, a continually blinking light on a static background elicits sustained flicker due to its locally outlier temporal dynamics but is only surprising for a moment. Similarly, a shower of randomly-colored pixels continually excites all low-level feature detectors but rapidly becomes unsurprising.

**Strongest attractors of human attention.** Clearly, in our and previous eye-tracking experiments, in some situations potentially interesting targets were more numerous than in others. With many possible targets, different observers may orient towards different locations, making it more difficult for a single metric to accurately predict all observers. Hence we consider (Figure 4) subsets of human saccades where at least two, three, or all four observers simultaneously agreed on a gaze target. Observers could have agreed based on bottom-up factors (e.g., only one location had interesting visual appearance at that time), top-down factors (e.g., only one object was of current cognitive interest), or both (e.g., a single cognitively interesting object was present which also had distinctive appearance). Irrespectively of the cause for agreement, it indicates consolidated belief that a location was attractive. While the $KL$ scores of all metrics improved when progressively focusing onto only those locations, dynamic metrics improved more steeply, indicating that stimuli which more reliably attracted all observers carried more motion, saliency, and surprise. Surprise remained significantly the best metric to characterize these agreed-upon attractors of human gaze ($p < 10^{-100}$ or better on $t$-tests for equality of $KL$ scores).

Overall, surprise explained the greatest fraction of human saccades, indicating that humans are significantly attracted towards surprising locations in video displays. Over 72% of all human saccades were targeted to locations predicted to be more surprising than on average. When only considering saccades where two, three, or four observers agreed on a common gaze target, this figure rose to 76%, 80%, and 84%, respectively.

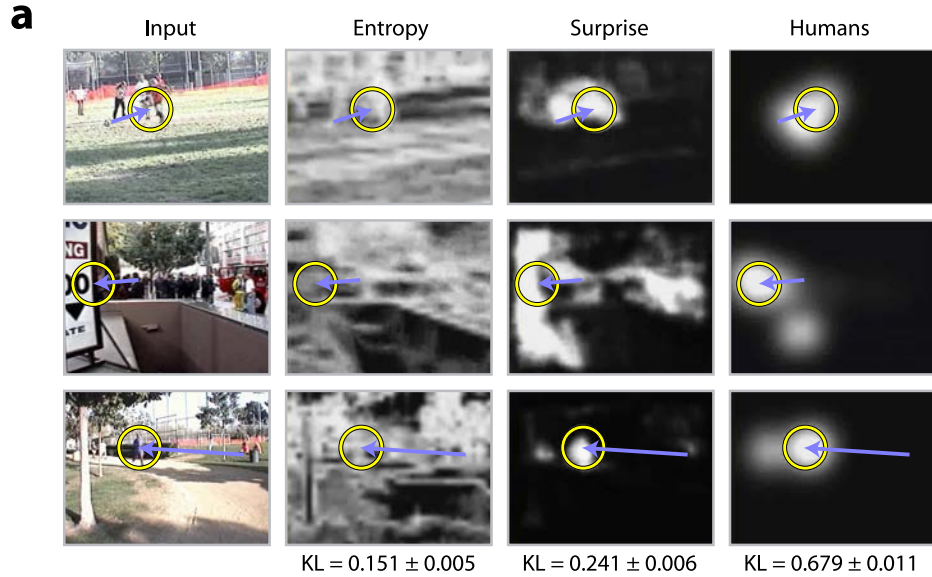

KL = 0.151 ± 0.005    KL = 0.241 ± 0.006    KL = 0.679 ± 0.011

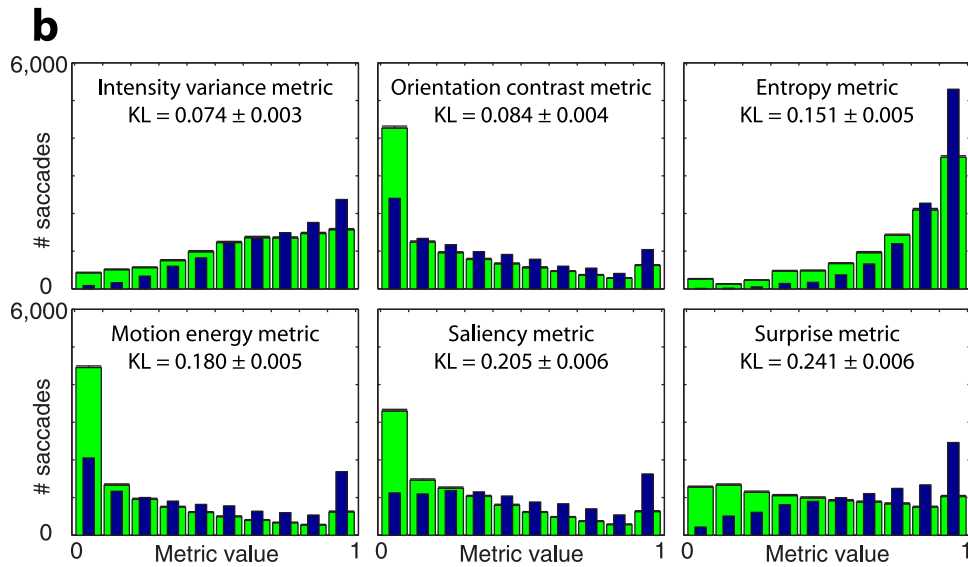

Figure 3: **(a)** Sample video frames, with corresponding human saccades and predictions from the entropy, surprise, and human-derived metrics. Entropy maps, like intensity variance and orientation maps, exhibited many locations with high responses, hence had low specificity and were poorly discriminative. In contrast, motion, saliency, and surprise maps were much sparser and more specific, with surprise significantly more often on target. For three example frames (first column), saccades from one subject are shown (arrows) with corresponding apertures over which master map activity at the saccade endpoint was sampled (circles). **(b)** $KL$ scores for these metrics indicate significantly different performance levels, and a strict ranking of variance < orientation < entropy < motion < saliency < surprise < human-derived. $KL$ scores were computed by comparing the number of human saccades landing onto each given range of master map values (narrow blue bars) to the number of random saccades hitting the same range (wider green bars). A score of zero would indicate equality between the human and random histograms, i.e., humans did not tend to hit various master map values any differently from expected by chance, or, the master map could not predict human saccades better than random saccades. Among the six computational metrics tested in total, surprise performed best, in that surprising locations were relatively few yet reliably gazed to by humans.

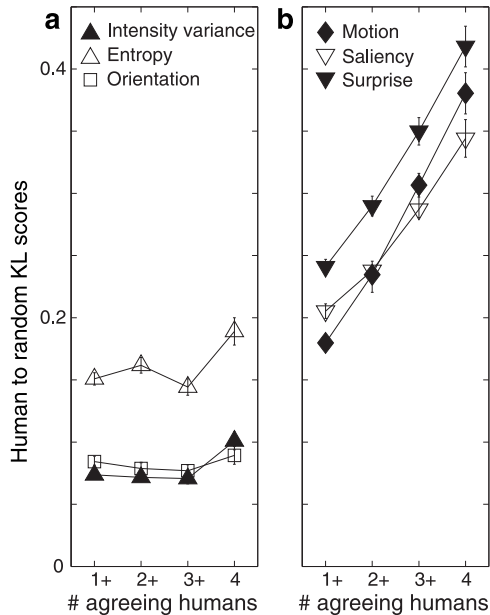

Figure 4: $KL$ scores when considering only saccades where at least one (all 10,192 saccades), two (7,948 saccades), three (5,565 saccades), or all four (2,951 saccades) humans agreed on a common gaze location, for the static **(a)** and dynamic metrics **(b)**. Static metrics improved substantially when progressively focusing onto saccades with stronger inter-observer agreement (average slope $0.56 \pm 0.37$ percent $KL$ score units per 1,000 pruned saccades). Hence, when humans agreed on a location, they also tended to be more reliably predicted by the metrics. Furthermore, dynamic metrics improved 4.5 times more steeply (slope $2.44 \pm 0.37$), suggesting a stronger role of dynamic events in attracting human attention. Surprising events were significantly the strongest ($t$-tests for equality of $KL$ scores between surprise and other metrics, $p < 10^{-100}$).

## 3 Discussion

While previous research has shown with either static scenes or dynamic synthetic stimuli that humans preferentially fixate regions of high entropy [34], contrast [31], saliency [32], flicker [36], or motion [37], our data provides direct experimental evidence that humans fixate surprising locations even more reliably. These conclusions were made possible by developing new tools to quantify what attracts human gaze over space and time in dynamic natural scenes. Surprise explained best where humans look when considering all saccades, and even more so when restricting the analysis to only those saccades for which human observers tended to agree. Surprise hence represents an inexpensive, easily computable approximation to human attentional allocation.

In the absence of quantitative tools to measure surprise, most experimental and modeling work to date has adopted the approximation that novel events are surprising, and has focused on experimental scenarios which are simple enough to ensure an overlap between informal notions of novelty and surprise: for example, a stimulus is novel during testing if it has not been seen during training [9]. Our definition opens new avenues for more sophisticated experiments, where surprise elicited by different stimuli can be precisely compared and calibrated, yielding predictions at the single-unit as well as behavioral levels.

The definition of surprise — as the distance between the posterior and prior distributions of beliefs over models — is entirely general and readily applicable to the analysis of auditory, olfactory, gustatory, or somatosensory data. While here we have focused on behavior rather than detailed biophysical implementation, it is worth noting that detecting surprise in neural spike trains does not require semantic understanding of the data carried by the spike trains, and thus could provide guiding signals during self-organization and development of sensory areas. At higher processing levels, top-down cues and task demands are known to combine with stimulus novelty in capturing attention and triggering learning [1, 38], ideas which may now be formalized and quantified in terms of priors, posteriors, and surprise. Surprise, indeed, inherently depends on uncertainty and on prior beliefs. Hence surprise theory can further be tested and utilized in experiments where the prior is biased, for ex-

ample by top-down instructions or prior exposures to stimuli [38]. In addition, simple surprise-based behavioral measures such as the eye-tracking one used here may prove useful for early diagnostic of human conditions including autism and attention-deficit hyperactive disorder, as well as for quantitative comparison between humans and animals which may have lower or different priors, including monkeys, frogs, and flies. Beyond sensory biology, computable surprise could guide the development of data mining and compression systems (giving more bits to surprising regions of interest), to find surprising agents in crowds, surprising sentences in books or speeches, surprising sequences in genomes, surprising medical symptoms, surprising odors in airport luggage racks, surprising documents on the world-wide-web, or to design surprising advertisements.

**Acknowledgments:** *Supported by HFSP, NSF and NGA (L.I.), NIH and NSF (P.B.). We thank UCI's Institute for Genomics and Bioinformatics and USC's Center High Performance Computing and Communications (www.usc.edu/hpcc) for access to their computing clusters.*

# References

[1] Ranganath, C. & Rainer, G. *Nat Rev Neurosci* **4**, 193–202 (2003).

[2] Rao, R. P. & Ballard, D. H. *Nat Neurosci* **2**, 79–87 (1999).

[3] Olshausen, B. A. & Field, D. J. *Nature* **381**, 607–609 (1996).

[4] Müller, J. R., Metha, A. B., Krauskopf, J. & Lennie, P. *Science* **285**, 1405–1408 (1999).

[5] Dragoi, V., Sharma, J., Miller, E. K. & Sur, M. *Nat Neurosci* **5**, 883–891 (2002).

[6] David, S. V., Vinje, W. E. & Gallant, J. L. *J Neurosci* **24**, 6991–7006 (2004).

[7] Maffei, L., Fiorentini, A. & Bisti, S. *Science* **182**, 1036–1038 (1973).

[8] Movshon, J. A. & Lennie, P. *Nature* **278**, 850–852 (1979).

[9] Fecteau, J. H. & Munoz, D. P. *Nat Rev Neurosci* **4**, 435–443 (2003).

[10] Kurahashi, T. & Menini, A. *Nature* **385**, 725–729 (1997).

[11] Bradley, J., Bonigk, W., Yau, K. W. & Frings, S. *Nat Neurosci* **7**, 705–710 (2004).

[12] Ulanovsky, N., Las, L. & Nelken, I. *Nat Neurosci* **6**, 391–398 (2003).

[13] Solomon, S. G., Peirce, J. W., Dhruv, N. T. & Lennie, P. *Neuron* **42**, 155–162 (2004).

[14] Smirnakis, S. M., Berry, M. J. & et al. *Nature* **386**, 69–73 (1997).

[15] Brown, S. P. & Masland, R. H. *Nat Neurosci* **4**, 44–51 (2001).

[16] Kennedy, H. J., Evans, M. G. & et al. *Nat Neurosci* **6**, 832–836 (2003).

[17] Schultz, W. & Dickinson, A. *Annu Rev Neurosci* **23**, 473–500 (2000).

[18] Fletcher, P. C., Anderson, J. M., Shanks, D. R. et al. *Nat Neurosci* **4**, 1043–1048 (2001).

[19] Knight, R. *Nature* **383**, 256–259 (1996).

[20] Stern, C. E., Corkin, S., Gonzalez, R. G. et al. *Proc Natl Acad Sci U S A* **93**, 8660–8665 (1996).

[21] Li, S., Cullen, W. K., Anwyl, R. & Rowan, M. J. *Nat Neurosci* **6**, 526–531 (2003).

[22] Ebstein, R. P., Novick, O., Umansky, R. et al. *Nat Genet* **12**, 78–80 (1996).

[23] Benjamin, J., Li, L. & et al. *Nat Genet* **12**, 81–84 (1996).

[24] Lusher, J. M., Chandler, C. & Ball, D. *Mol Psychiatry* **6**, 497–499 (2001).

[25] Itti, L. & Baldi, P. In *Proc. IEEE CVPR*. San Siego, CA (2005 in press).

[26] Cox, R. T. *Am. J. Phys.* **14**, 1–13 (1964).

[27] Savage, L. J. *The foundations of statistics* (Dover, New York, 1972). (First Edition in 1954).

[28] Jaynes, E. T. *Probability Theory. The Logic of Science* (Cambridge University Press, 2003).

[29] Kullback, S. *Information Theory and Statistics* (Wiley, New York:New York, 1959).

[30] Itti, L. *Visual Cognition* (2005 in press).

[31] Reinagel, P. & Zador, A. M. *Network* **10**, 341–350 (1999).

[32] Parkhurst, D., Law, K. & Niebur, E. *Vision Res* **42**, 107–123 (2002).

[33] Itti, L. & Koch, C. *Nat Rev Neurosci* **2**, 194–203 (2001).

[34] Privitera, C. M. & Stark, L. W. *IEEE Trans Patt Anal Mach Intell* **22**, 970–982 (2000).

[35] All source code for all metrics is freely available at http://iLab.usc.edu/toolkit/.

[36] Theeuwes, J. *Percept Psychophys* **57**, 637–644 (1995).

[37] Abrams, R. A. & Christ, S. E. *Psychol Sci* **14**, 427–432 (2003).

[38] Wolfe, J. M. & Horowitz, T. S. *Nat Rev Neurosci* **5**, 495–501 (2004).
